# Active Bidirectional Coupling in a Cochlear Chip

**Bo Wen and Kwabena Boahen**
Department of Bioengineering
University of Pennsylvania
Philadelphia, PA 19104
{wenbo,boahen}@seas.upenn.edu

## Abstract

We present a novel cochlear model implemented in analog very large scale integration (VLSI) technology that emulates nonlinear active cochlear behavior. This silicon cochlea includes outer hair cell (OHC) electromotility through active bidirectional coupling (ABC), a mechanism we proposed in which OHC motile forces, through the microanatomical organization of the organ of Corti, realize the cochlear amplifier. Our chip measurements demonstrate that frequency responses become larger and more sharply tuned when ABC is turned on; the degree of the enhancement decreases with input intensity as ABC includes saturation of OHC forces.

## 1 Silicon Cochleae

Cochlear models, mathematical and physical, with the shared goal of emulating nonlinear active cochlear behavior, shed light on how the cochlea works if based on cochlear micromechanics. Among the modeling efforts, silicon cochleae have promise in meeting the need for real-time performance and low power consumption. Lyon and Mead developed the first analog electronic cochlea [1], which employed a cascade of second-order filters with exponentially decreasing resonant frequencies. However, the cascade structure suffers from delay and noise accumulation and lacks fault-tolerance. Modeling the cochlea more faithfully, Watts built a two-dimensional (2D) passive cochlea that addressed these shortcomings by incorporating the cochlear fluid using a resistive network [2]. This parallel structure, however, has its own problem: response gain is diminished by interference among the second-order sections' outputs due to the large phase change at resonance [3].

Listening more to biology, our silicon cochlea aims to overcome the shortcomings of existing architectures by mimicking the cochlear micromechanics while including outer hair cell (OHC) electromotility. Although how exactly OHC motile forces boost the basilar membrane's (BM) vibration remains a mystery, cochlear microanatomy provides clues. Based on these clues, we previously proposed a novel mechanism, active bidirectional coupling (ABC), for the cochlear amplifier [4]. Here, we report an analog VLSI chip that implements this mechanism. In essence, our implementation is the first silicon cochlea that employs stimulus enhancement (i.e., active behavior) instead of undamping (i.e., high filter $Q$ [5]).

The paper is organized as follows. In Section 2, we present the hypothesized mechanism (ABC), first described in [4]. In Section 3, we provide a mathematical formulation of the

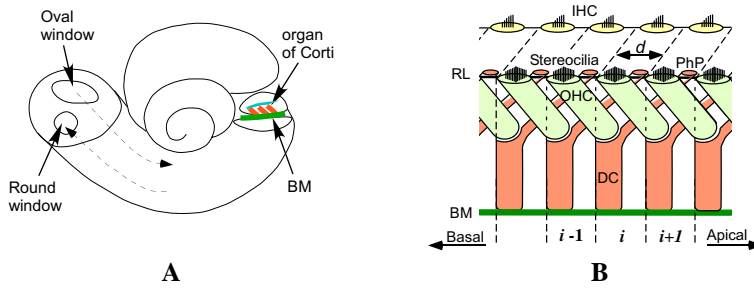

Figure 1: The inner ear. **A** Cutaway showing cochlear ducts (adapted from [6]). **B** Longitudinal view of cochlear partition (CP) (modified from [7]-[8]). Each outer hair cell (OHC) tilts toward the base while the Deiter's cell (DC) on which it sits extends a phalangeal process (PhP) toward the apex. The OHCs' stereocilia and the PhPs' apical ends form the reticular lamina (RL). $d$ is the tilt distance, and the segment size. IHC: inner hair cell.

model as the basis of cochlear circuit design. Then we proceed in Section 4 to synthesize the circuit for the cochlear chip. Last, we present chip measurements in Section 5 that demonstrate nonlinear active cochlear behavior.

## 2 Active Bidirectional Coupling

The cochlea actively amplifies acoustic signals as it performs spectral analysis. The movement of the stapes sets the cochlear fluid into motion, which passes the stimulus energy onto a certain region of the BM, the main vibrating organ in the cochlea (Figure 1A). From the base to the apex, BM fibers increase in width and decrease in thickness, resulting in an exponential decrease in stiffness which, in turn, gives rise to the passive frequency tuning of the cochlea. The OHCs' electromotility is widely thought to account for the cochlea's exquisite sensitivity and discriminability. The exact way that OHC motile forces enhance the BM's motion, however, remains unresolved.

We propose that the triangular mechanical unit formed by an OHC, a phalangeal process (PhP) extended from the Deiter's cell (DC) on which the OHC sits, and a portion of the reticular lamina (RL), between the OHC's stereocilia end and the PhP's apical tip, plays an active role in enhancing the BM's responses (Figure 1B). The cochlear partition (CP) is divided into a number of segments longitudinally. Each segment includes one DC, one PhP's apical tip and one OHC's stereocilia end, both attached to the RL. Approximating the anatomy, we assume that when an OHC's stereocilia end lies in segment $i - 1$, its basolateral end lies in the immediately apical segment $i$. Furthermore, the DC in segment $i$ extends a PhP that angles toward the apex of the cochlea, with its apical end inserted just behind the stereocilia end of the OHC in segment $i + 1$.

Our hypothesis (ABC) includes both feedforward and feedbackward interactions. On one hand, the feedforward mechanism, proposed in [9], hypothesized that the force resulting from OHC contraction or elongation is exerted onto an adjacent downstream BM segment due to the OHC's basal tilt. On the other hand, the novel insight of the feedbackward mechanism is that the OHC force is delivered onto an adjacent upstream BM segment due to the apical tilt of the PhP extending from the DC's main trunk.

In a nutshell, the OHC motile forces, through the microanatomy of the CP, feed forward and backward, in harmony with each other, resulting in bidirectional coupling between BM segments in the longitudinal direction. Specifically, due to the opposite action of OHC

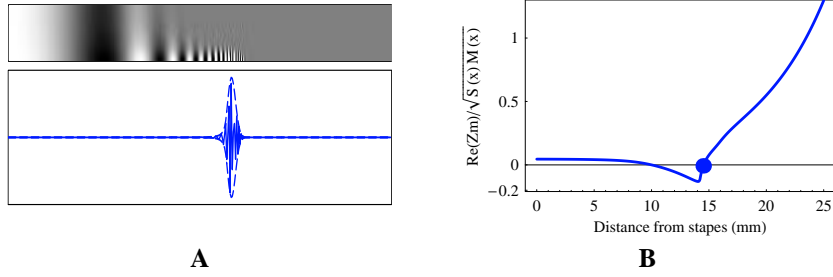

Figure 2: Wave propagation (WP) and basilar membrane (BM) impedance in the active cochlear model with a 2kHz pure tone ($\alpha = 0.15$, $\gamma = 0.3$). **A** WP in fluid and BM. **B** BM impedance $Z_{\mathrm{m}}$ (i.e., pressure divided by velocity), normalized by $\sqrt{S(x)M(x)}$. Only the resistive component is shown; dot marks peak location.

forces on the BM and the RL, the motion of BM segment $i-1$ reinforces that of segment $i$ while the motion of segment $i+1$ opposes that of segment $i$, as described in detail in [4].

## 3 The 2D Nonlinear Active Model

To provide a blueprint for the cochlear circuit design, we formulate a 2D model of the cochlea that includes ABC. Both the cochlea's length (BM) and height (cochlear ducts) are discretized into a number of segments, with the original aspect ratio of the cochlea maintained. In the following expressions, $x$ represents the distance from the stapes along the CP, with $x = 0$ at the base (or the stapes) and $x = L$ (uncoiled cochlear duct length) at the apex; $y$ represents the vertical distance from the BM, with $y = 0$ at the BM and $y = \pm h$ (cochlear duct radius) at the bottom/top wall.

Providing that the assumption of fluid incompressibility holds, the velocity potential $\phi$ of the fluids is required to satisfy $\bigtriangledown^2 \phi(x, y, t) = 0$, where $\bigtriangledown^2$ denotes the Laplacian operator. By definition, this potential is related to fluid velocities in the $x$ and $y$ directions: $V_x = -\partial\phi/\partial x$ and $V_y = -\partial\phi/\partial y$.

The BM is driven by the fluid pressure difference across it. Hence, the BM's vertical motion (with downward displacement being positive) can be described as follows.

$$P_{\mathrm{d}}(x) + F_{\mathrm{OHC}}(x) = S(x)\delta(x) + \beta(x)\dot{\delta}(x) + M(x)\ddot{\delta}(x), \qquad (1)$$

where $S(x)$ is the stiffness, $\beta(x)$ is the damping, and $M(x)$ is the mass, per unit area, of the BM; $\delta$ is the BM's downward displacement. $P_{\mathrm{d}} = \rho\,\partial(\phi_{\mathrm{SV}}(x, y, t) - \phi_{\mathrm{ST}}(x, y, t))/\partial t$ is the pressure difference between the two fluid ducts (the scala vestibuli (SV) and the scala tympani (ST)), evaluated at the BM ($y = 0$); $\rho$ is the fluid density.

The $F_{\mathrm{OHC}}(x)$ term combines feedforward and feedbackward OHC forces, described by

$$F_{\mathrm{OHC}}(x) = s_0\big(\tanh(\alpha\gamma S(x)\delta(x-d)/s_0) - \tanh(\alpha S(x)\delta(x+d)/s_0)\big), \qquad (2)$$

where $\alpha$ denotes the OHC motility, expressed as a fraction of the BM stiffness, and $\gamma$ is the ratio of feedforward to feedbackward coupling, representing relative strengths of the OHC forces exerted on the BM segment through the DC, directly and via the tilted PhP. $d$ denotes the tilt distance, which is the horizontal displacement between the source and the recipient of the OHC force, assumed to be equal for the forward and backward cases. We use the hyperbolic tangent function to model saturation of the OHC forces, the nonlinearity that is evident in physiological measurements [8]; $s_0$ determines the saturation level.

We observed wave propagation in the model and computed the BM's impedance (i.e., the ratio of driving pressure to velocity). Following the semi-analytical approach in [2], we simulated a linear version of the model (without saturation). The traveling wave transitions from long-wave to short-wave before the BM vibration peaks; the wavelength around the characteristic place is comparable to the tilt distance (Figure 2A). The BM impedance's real part (i.e., the resistive component) becomes negative before the peak (Figure 2B). On the whole, inclusion of OHC motility through ABC boosts the traveling wave by pumping energy onto the BM when the wavelength matches the tilt of the OHC and PhP.

## 4  Analog VLSI Design and Implementation

Based on our mathematical model, which produces realistic responses, we implemented a 2D nonlinear active cochlear circuit in analog VLSI, taking advantage of the 2D nature of silicon chips. We first synthesize a circuit analog of the mathematical model, and then we implement the circuit in the log-domain. We start by synthesizing a passive model, and then extend it to a nonlinear active one by including ABC with saturation.

### 4.1  Synthesizing the BM Circuit

The model consists of two fundamental parts: the cochlear fluid and the BM. First, we design the fluid element and thus the fluid network. In discrete form, the fluids can be viewed as a grid of elements with a specific resistance that corresponds to the fluid density or mass. Since charge is conserved for a small sheet of resistance and so are particles for a small volume of fluid, we use current to simulate fluid velocity. At the transistor level, the current flowing through the channel of a MOS transistor, operating subthreshold as a diffusive element, can be used for this purpose. Therefore, following the approach in [10], we implement the cochlear fluid network using a diffusor network formed by a 2D grid of nMOS transistors.

Second, we design the BM element and thus the BM. As current represents velocity, we rewrite the BM boundary condition (Equation 1, without the $F_{\mathrm{OHC}}$ term):

$$\dot{I}_{\mathrm{in}} = S(x) \int I_{\mathrm{mem}} dt + \beta(x) I_{\mathrm{mem}} + M(x) \dot{I}_{\mathrm{mem}}, \tag{3}$$

where $I_{\mathrm{in}}$, obtained by applying the voltage from the diffusor network to the gate of a pMOS transistor, represents the velocity potential scaled by the fluid density. In turn, $I_{\mathrm{mem}}$ drives the diffusor network to match the fluid velocity with the BM velocity, $\dot{\delta}$. The $F_{\mathrm{OHC}}$ term is dealt with in Section 4.2.

Implementing this second-order system requires two state-space variables, which we name $I_{\mathrm{s}}$ and $I_{\mathrm{o}}$. And with $s = j\omega$, our synthesized BM design (passive) is

$$\tau_1 I_{\mathrm{s}} s + I_{\mathrm{s}} = -I_{\mathrm{in}} + I_{\mathrm{o}}, \tag{4}$$

$$\tau_2 I_{\mathrm{o}} s + I_{\mathrm{o}} = I_{\mathrm{in}} - b I_{\mathrm{s}}, \tag{5}$$

$$I_{\mathrm{mem}} = I_{\mathrm{in}} + I_{\mathrm{s}} - I_{\mathrm{o}}, \tag{6}$$

where the two first-order systems are both low-pass filters (LPFs), with time constants $\tau_1$ and $\tau_2$, respectively; $b$ is a gain factor. Thus, $I_{\mathrm{in}}$ can be expressed in terms of $I_{\mathrm{mem}}$ as:

$$I_{\mathrm{in}} s^2 = \left( (b+1)/\tau_1\tau_2 + ((\tau_1 + \tau_2)/\tau_1\tau_2)s + s^2 \right) I_{\mathrm{mem}}.$$

Comparing this expression with the design target (Equation 3) yields the circuit analogs:

$$S(x) = (b+1)/\tau_1\tau_2, \quad \beta(x) = (\tau_1 + \tau_2)/\tau_1\tau_2, \quad \text{and} \quad M(x) = 1.$$

Note that the mass $M(x)$ is a constant (i.e., 1), which was also the case in our mathematical model simulation. These analogies require that $\tau_1$ and $\tau_2$ increase exponentially to

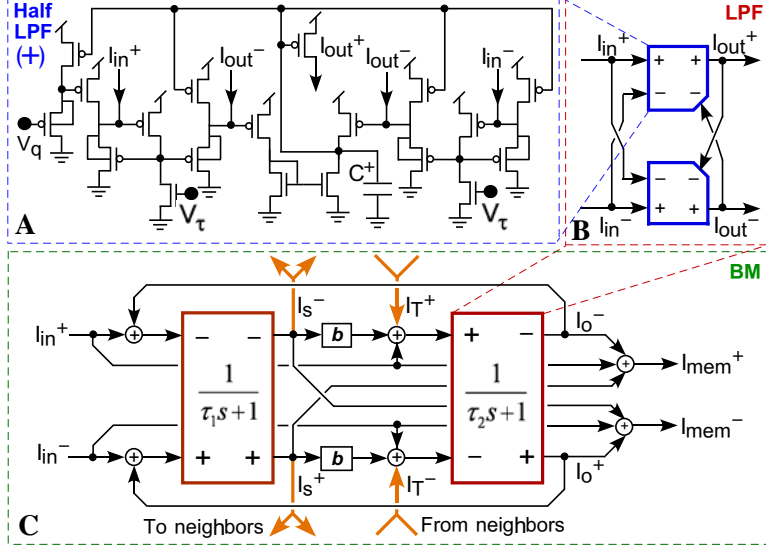

Figure 3: Low-pass filter (LPF) and second-order section circuit design. **A** Half-LPF circuit. **B** Complete LPF circuit formed by two half-LPF circuits. **C** Basilar membrane (BM) circuit. It consists of two LPFs and connects to its neighbors through $I_s$ and $I_T$.

simulate the exponentially decreasing BM stiffness (and damping); $b$ allows us to achieve a reasonable stiffness for a practical choice of $\tau_1$ and $\tau_2$ (capacitor size is limited by silicon area).

## 4.2 Adding Active Bidirectional Coupling

To include ABC in the BM boundary condition, we replace $\delta$ in Equation 2 with $\int I_{\mathrm{mem}} dt$ to obtain

$$F_{\mathrm{OHC}} = r_{\mathrm{ff}} S(x) \mathcal{T}\big(\int I_{\mathrm{mem}}(x-d) dt\big) - r_{\mathrm{fb}} S(x) \mathcal{T}\big(\int I_{\mathrm{mem}}(x+d) dt\big),$$

where $r_{\mathrm{ff}} = \alpha\gamma$ and $r_{\mathrm{fb}} = \alpha$ denote the feedforward and feedbackward OHC motility factors, and $\mathcal{T}$ denotes saturation. The saturation is applied to the displacement, instead of the force, as this simplifies the implementation. We obtain the integrals by observing that, in the passive design, the state variable $I_s = -I_{\mathrm{mem}}/s\tau_1$. Thus, $\int I_{\mathrm{mem}}(x-d) dt = -\tau_{1\mathrm{f}} I_{\mathrm{sf}}$ and $\int I_{\mathrm{mem}}(x+d) dt = -\tau_{1\mathrm{b}} I_{\mathrm{sb}}$. Here, $I_{\mathrm{sf}}$ and $I_{\mathrm{sb}}$ represent the outputs of the first LPF in the upstream and downstream BM segments, respectively; $\tau_{1\mathrm{f}}$ and $\tau_{1\mathrm{b}}$ represent their respective time constants. To reduce complexity in implementation, we use $\tau_1$ to approximate both $\tau_{1\mathrm{f}}$ and $\tau_{1\mathrm{b}}$ as the longitudinal span is small.

We obtain the active BM design by replacing Equation 5 with the synthesis result:

$$\tau_2 I_{\mathrm{o}} s + I_{\mathrm{o}} = I_{\mathrm{in}} - b I_s + r_{\mathrm{fb}}(b+1)\mathcal{T}(-I_{\mathrm{sb}}) - r_{\mathrm{ff}}(b+1)\mathcal{T}(-I_{\mathrm{sf}}).$$

Note that, to implement ABC, we only need to add two currents to the second LPF in the passive system. These currents, $I_{\mathrm{sf}}$ and $I_{\mathrm{sb}}$, come from the upstream and downstream neighbors of each segment.

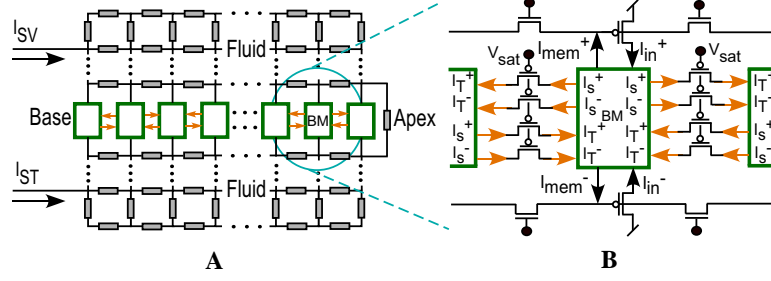

Figure 4: Cochlear chip. **A** Architecture: Two diffusive grids with embedded BM circuits model the cochlea. **B** Detail. BM circuits exchange currents with their neighbors.

### 4.3 Class AB Log-domain Implementation

We employ the log-domain filtering technique [11] to realize current-mode operation. In addition, following the approach proposed in [12], we implement the circuit in Class AB to increase dynamic range, reduce the effect of mismatch and lower power consumption. This differential signaling is inspired by the way the biological cochlea works—the vibration of BM is driven by the pressure difference across it.

Taking a bottom-up strategy, we start by designing a Class AB LPF, a building block for the BM circuit. It is described by

$$\tau(I_{\text{out}}^+ - I_{\text{out}}^-)s + (I_{\text{out}}^+ - I_{\text{out}}^-) = I_{\text{in}}^+ - I_{\text{in}}^- \ \text{ and } \ \tau I_{\text{out}}^+ I_{\text{out}}^- s + I_{\text{out}}^+ I_{\text{out}}^- = I_{\text{q}}^2,$$

where $I_{\text{q}}$ sets the geometric mean of the positive and negative components of the output current, and $\tau$ sets the time constant.

Combining the common-mode constraint with the differential design equation yields the nodal equation for the positive path (the negative path has superscripts $+$ and $-$ swapped):

$$C\dot{V}_{\text{out}}^+ = I_\tau \left( (I_{\text{in}}^+ - I_{\text{in}}^-) + (I_{\text{q}}^2/I_{\text{out}}^+ - I_{\text{out}}^+) \right) / (I_{\text{out}}^+ + I_{\text{out}}^-).$$

This nodal equation suggests the half-LPF circuit shown in Figure 3A. $V_{\text{out}}^+$, the voltage on the positive capacitor ($C^+$), gates a pMOS transistor to produce the corresponding current signal, $I_{\text{out}}^+$ ($V_{\text{out}}^-$ and $I_{\text{out}}^-$ are similarly related). The bias $V_{\text{q}}$ sets the quiescent current $I_{\text{q}}$ while $V_\tau$ determines the current $I_\tau$, which is related to the time constant by $\tau = C\mathrm{u_T}/\kappa I_\tau$ ($\kappa$ is the subthreshold slope coefficient and $\mathrm{u_T}$ is the thermal voltage). Two of these sub-circuits, connected in push–pull, form a complete LPF (Figure 3B).

The BM circuit is implemented using two LPFs interacting in accordance with the synthesized design equations (Figure 3C). $I_{\text{mem}}$ is the combination of three currents, $I_{\text{in}}$, $I_{\text{s}}$, and $I_{\text{o}}$. Each BM sends out $I_{\text{s}}$ and receives $I_{\text{T}}$, a saturated version of its neighbor's $I_{\text{s}}$. The saturation is accomplished by a current-limiting transistor (see Figure 4B), which yields $I_{\text{T}} = \mathcal{T}(I_{\text{s}}) = I_{\text{s}}I_{\text{sat}}/(I_{\text{s}} + I_{\text{sat}})$, where $I_{\text{sat}}$ is set by a bias voltage $V_{\text{sat}}$.

### 4.4 Chip Architecture

We fabricated a version of our cochlear chip architecture (Figure 4) with 360 BM circuits and two 4680-element fluid grids ($360 \times 13$). This chip occupies $10.9\mathrm{mm}^2$ of silicon area in $0.25\mu$m CMOS technology. Differential input signals are applied at the base while the two fluid grids are connected at the apex through a fluid element that represents the helicotrema.

## 5    Chip Measurements

We carried out two measurements that demonstrate the desired amplification by ABC, and the compressive growth of BM responses due to saturation. To obtain sinusoidal current as the input to the BM subcircuits, we set the voltages applied at the base to be the logarithm of a half-wave rectified sinusoid.

We first investigated BM-velocity frequency responses at six linearly spaced cochlear positions (Figure 5). The frequency that maximally excites the first position (Stage 30), defined as its characteristic frequency (CF), is 12.1kHz. The remaining five CFs, from early to later stages, are 8.2k, 1.7k, 905, 366, and 218Hz, respectively. Phase accumulation at the CFs ranges from 0.56 to $2.67\pi$ radians, comparable to $1.67\pi$ radians in the mammalian cochlea [13]. $Q_{10}$ factor (the ratio of the CF to the bandwidth 10dB below the peak) ranges from 1.25 to 2.73, comparable to 2.55 at mid-sound intensity in biology (computed from [13]). The cutoff slope ranges from -20 to -54dB/octave, as compared to -85dB/octave in biology (computed from [13]).

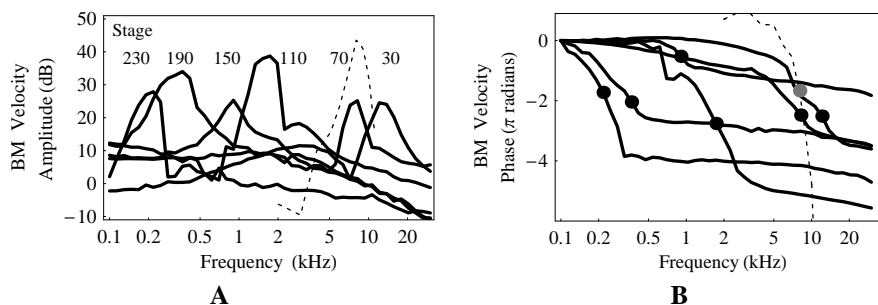

Figure 5:    Measured BM-velocity frequency responses at six locations.    **A** Amplitude.
**B** Phase. Dashed lines: Biological data (adapted from [13]). Dots mark peaks.

We then explored the longitudinal pattern of BM-velocity responses and the effect of ABC. Stimulating the chip using four different pure tones, we obtained responses in which a 4kHz input elicits a peak around Stage 85 while 500Hz sound travels all the way to Stage 178 and peaks there (Figure 6A). We varied the input voltage level and obtained frequency responses at Stage 100 (Figure 6B). Input voltage level increases linearly such that the current increases exponentially; the input current level (in dB) was estimated based on the measured $\kappa$ for this chip. As expected, we observed linearly increasing responses at low frequencies in the logarithmic plot. In contrast, the responses around the CF increase less and become broader with increasing input level as saturation takes effect in that region (resembling a passive cochlea). We observed 24dB compression as compared to 27 to 47dB in biology [13]. At the highest intensities, compression also occurs at low frequencies.

These chip measurements demonstrate that inclusion of ABC, simply through coupling neighboring BM elements, transforms a passive cochlea into an active one. This active cochlear model's nonlinear responses are qualitatively comparable to physiological data.

## 6    Conclusions

We presented an analog VLSI implementation of a 2D nonlinear cochlear model that utilizes a novel active mechanism, ABC, which we proposed to account for the cochlear amplifier. ABC was shown to pump energy into the traveling wave. Rather than detecting the wave's amplitude and implementing an automatic-gain-control loop, our biomorphic model accomplishes this simply by nonlinear interactions between adjacent neighbors. Im-

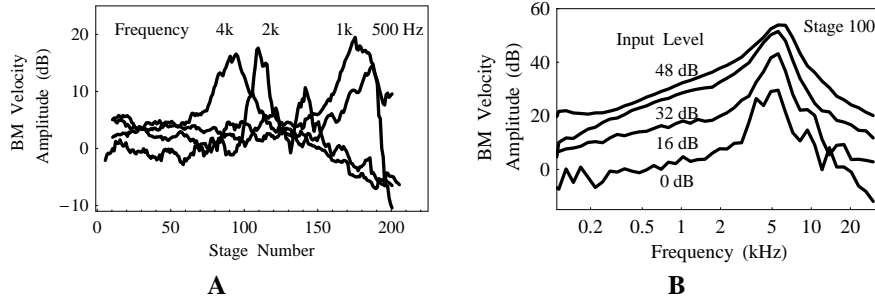

Figure 6: Measured BM-velocity responses (cont'd). **A** Longitudinal responses (20-stage moving average). Peak shifts to earlier (basal) stages as input frequency increases from 500 to 4kHz. **B** Effects of increasing input intensity. Responses become broader and show compressive growth.

plemented in the log-domain, with Class AB operation, our silicon cochlea shows enhanced frequency responses, with compressive behavior around the CF, when ABC is turned on. These features are desirable in prosthetic applications and automatic speech recognition systems as they capture the properties of the biological cochlea.

## References

[1] Lyon, R.F. & Mead, C.A. (1988) An analog electronic cochlea. *IEEE Trans. Acoust. Speech and Signal Proc.*, **36**: 1119-1134.

[2] Watts, L. (1993) *Cochlear Mechanics: Analysis and Analog VLSI*. Ph.D. thesis, Pasadena, CA: California Institute of Technology.

[3] Fragnière, E. (2005) A 100-Channel analog CMOS auditory filter bank for speech recognition. *IEEE International Solid-State Circuits Conference (ISSCC 2005)*, pp. 140-141.

[4] Wen, B. & Boahen, K. (2003) A linear cochlear model with active bi-directional coupling. *The 25th Annual International Conference of the IEEE Engineering in Medicine and Biology Society (EMBC 2003)*, pp. 2013-2016.

[5] Sarpeshkar, R., Lyon, R.F., & Mead, C.A. (1996) An analog VLSI cochlear model with new transconductance amplifier and nonlinear gain control. *Proceedings of the IEEE Symposium on Circuits and Systems (ISCAS 1996)*, **3**: 292-295.

[6] Mead, C.A. (1989) *Analog VLSI and Neural Systems*. Reading, MA: Addison-Wesley.

[7] Russell, I.J. & Nilsen, K.E. (1997) The location of the cochlear amplifier: Spatial representation of a single tone on the guinea pig basilar membrane. *Proc. Natl. Acad. Sci. USA*, **94**: 2660-2664.

[8] Geisler, C.D. (1998) *From sound to synapse: physiology of the mammalian ear*. Oxford University Press.

[9] Geisler, C.D. & Sang, C. (1995) A cochlear model using feed-forward outer-hair-cell forces. *Hearing Research*, **86**: 132-146.

[10] Boahen, K.A. & Andreou, A.G. (1992) A contrast sensitive silicon retina with reciprocal synapses. In Moody, J.E. and Lippmann, R.P. (eds.), *Advances in Neural Information Processing Systems 4 (NIPS 1992)*, pp. 764-772, Morgan Kaufmann, San Mateo, CA.

[11] Frey, D.R. (1993) Log-domain filtering: an approach to current-mode filtering. *IEE Proc. G, Circuits Devices Syst.*, **140** (6): 406-416.

[12] Zaghloul, K. & Boahen, K.A. (2005) An On-Off log-domain circuit that recreates adaptive filtering in the retina. *IEEE Transactions on Circuits and Systems I: Regular Papers*, **52** (1): 99-107.

[13] Ruggero, M.A., Rich, N.C., Narayan, S.S., & Robles, L. (1997) Basilar membrane responses to tones at the base of the chinchilla cochlea. *J. Acoust. Soc. Am.*, **101** (4): 2151-2163.
